# A Neural Network to Detect Homologies in Proteins

**Yoshua Bengio**
School of Computer Science
McGill University
Montreal, Canada H3A 2A7

**Samy Bengio**
Département d'Informatique
Université de Montreal

**Yannick Pouliot**
Department of Biology
McGill University
Montreal Neurological Institute

**Patrick Agin**
Département d'Informatique
Université de Montreal

## ABSTRACT

In order to detect the presence and location of immunoglobulin (Ig) domains from amino acid sequences we built a system based on a neural network with one hidden layer trained with back propagation. The program was designed to efficiently identify proteins exhibiting such domains, characterized by a few localized conserved regions and a low overall homology. When the National Biomedical Research Foundation (NBRF) NEW protein sequence database was scanned to evaluate the program's performance, we obtained very low rates of false negatives coupled with a moderate rate of false positives.

## 1  INTRODUCTION

Two amino acid sequences from proteins are homologous if they can be aligned so that many corresponding amino acids are identical or have similar chemical properties. Such subsequences (domains) often exhibit similar three dimensional structure. Furthemore, sequence similarity often results from common ancestors. Immunoglobulin (Ig) domains are sets of $\beta$-sheets bound

by cysteine bonds and with a characteristic tertiary structure. Such domains are found in many proteins involved in immune, cell adhesion and receptor functions. These proteins collectively form the immunoglobulin superfamily (for review, see Williams and Barclay, 1987). Members of the superfamily often possess several Ig domains. These domains are characterized by well-conserved groups of amino acids localized to specific subregions. Other residues outside of these regions are often poorly conserved, such that there is low overall homology between Ig domains, even though they are clearly members of the same superfamily.

Current search programs incorporating algorithms such as the Wilbur-Lipman algorithm (1983) or the Needleman-Wunsch algorithm (1970) and its modification by Smith and Waterman (1981) are ill-designed for detecting such domains because they implicitly consider each amino acid to be equally important. This is not the case for residues within domains such as the Ig domain, since only some amino acids are well conserved, while most are variable. One solution to this problem are search algorithms based upon the statistical occurrence of a residue at a particular position (Wang *et al.*, 1989; Gribskov *et al.*, 1987). The Profile Analysis set of programs published by the University of Wisconsin Genetics Computer Group (Devereux *et al.*, 1984) rely upon such an algorithm. Although Profile Analysis can be applied to search for domains (*c.f.* Blaschuk, Pouliot & Holland 1990), the output from these programs often suffers from a high rate of false negatives and positives. Variations in domain length are handled using the traditional method of penalties proportional to the number of gaps introduced, their length and their position. This approach entails a significant amount of spurious recognition if there is considerable variation in domain length to be accounted for.

We have chosen to address these problems by training a neural network to recognize accepted Ig domains. Perceptrons and various types of neural networks have been used previously in biological research with various degrees of success (cf. Stormo *et al.*, 1982; Qian and Sejnowski, 1988). Our results suggest that they are well suited for detecting relatively cryptic sequence patterns such as those which characterize Ig domains. Because the design and training procedure described below is relatively simple, network-based search programs constitute a valid solution to problems such as searching for proteins assembled from the duplication of a domain.

## 2  ALGORITHM, NETWORK DESIGN AND TRAINING

The network capitalizes upon data concerning the existence and localization of highly conserved groups of amino acids characteristic of the Ig domain. Its design is similar in several respects to neural networks we have used in the study of speech recognition (Bengio *et al.*, 1989). Four conserved subregions (designated P1-P4) of the Ig domain homology were identified. These roughly correspond to $\beta$-strands B, C, E and F, respectively, of the Ig domain (see also Williams and Barclay, 1988). Amino acids in these four groups are not necessarily all conserved, but for each subregion they show a distribution very different from the distribution generally observed elsewhere in these proteins. Hence the first and most important stage of the system learns about these joint distributions. The program scans proteins using a window of 5 residues.

The first stage of the system consists of a 2-layer feedforward neural network (5 × 20 inputs - 8 hidden - 4 outputs; see Figure 1) trained with back propagation (Rumelhart *et al.*, 1986). Better results were obtained for the recognition of these conserved regions with this architecture than without hidden layer (similar to a perceptron). The second stage evaluates, based upon the stream of outputs generated by the first stage, whether and where a region similar to the Ig domain has been detected. This stage currently uses a simple dynamic programming algorithm, in which constraints about order of subregions and distance between them are explicitly programmed. We force the recognizer to detect a sequence of high values (above a threshold) for the four conserved regions, in the correct order and such that the sum of the values obtained at the four recognized regions is greater than a certain threshold. Weak penalties are applied for violations of distance constraints between conserved subregions (e.g., distance between P1 and P2, P2 and P3, etc) based upon simple rules derived from our analysis of Ig domains. These rules have little impact if strong homologies are detected, such that the program easily handles the large variation in domain size exhibited by Ig domains. It was necessary to explicitly formulate these constraints given the low number of training examples as well as the assumption that the distance between groups is not a critical discriminating factor. We have assumed that inter-region subsequences probably do not significantly influence discrimination.

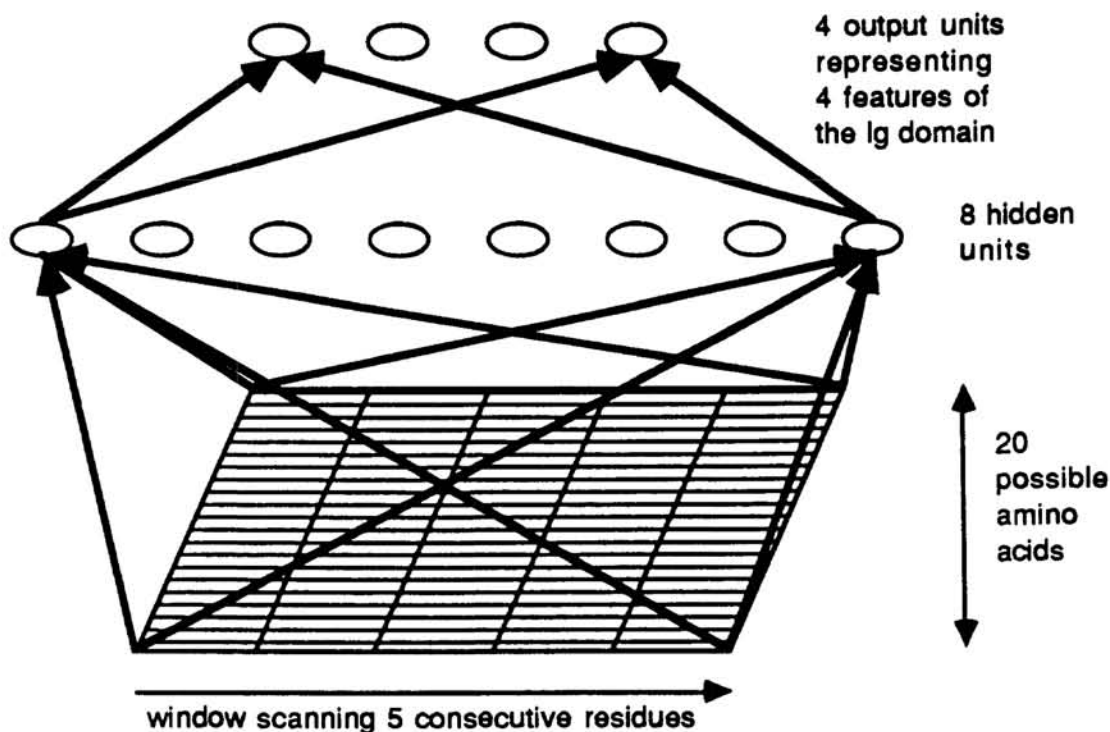

4 output units representing 4 features of the Ig domain

8 hidden units

20 possible amino acids

window scanning 5 consecutive residues

**Figure 1**: Structure of the neural network

```
filename : A22771.NEW
input sequence name : Ig epsilon chain C region - Human
HOMOLOGY starting at 24
VTLGCLATGYFPEPVMVTWDTGSLNGTTMTLPATTLTLSGHYATISLLTVSGAWAKQMFTC
   P1          P2          P3                              P4
Ending at 84. Score = 3.581
HOMOLOGY starting at 130
IQLLCLVSGYTPGTINITWLEDGQVMDVDLSTASTTQEGELASTQSELTLSQKHWLSDRTYTC
   P1          P2                              P3          P4
Ending at 192. Score = 3.825
HOMOLOGY starting at 234
PTITCLVVDLAPSKGTVNLTWSRASGKPVNHSTRKEEKQRNGTLTVTSTLPVGTRDWIEGETYQC
   P1          P2                          P3              P4
Ending at 298. Score = 3.351
HOMOLOGY starting at 340
RTLACLIQNFMPEDISVQWLHNEVQLPDARHSTTQPRKTKGSGFFVFSRLEVTRAEWEQKDEFIC
   P1          P2                          P3              P4
Ending at 404. Score = 3.402
```

**Figure 2:** Sample output from a search of NEW. Ig domains present within the constant region of an epsilon Ig chain (NBRF file number A22771) are listed with the position of P1-P4 (see text). The overall score for each domain is also listed.

As a training set we used a group of 30 proteins comprising bona fide Ig domains (Williams and Barclay, 1987). In order to increase the size of the training set, additional sequences were stochastically generated by substituting residues which are not in critical positions of the domain. These substitutions were designed not to affect the local distribution of residues to minimize changes in the overall chemical character of the region.

The program was evaluated and optimized by scanning the NBRF protein databases (PROTEIN and NEW) version 19. Results presented below are based upon searches of the NEW database (except where otherwise noted) and were generated with a cutoff value of 3.0. Only complete sequences from vertebrates, insects (including *Drosophila melanogaster*) and eukaryotic viruses were scanned. This corresponds to 2422 sequences out of the 4718 present in the NEW database. Trial runs with the program indicated that a cutoff threshold of between 2.7 and 3.0 eliminates the vast majority of false positives with little effect upon the rate of false negatives. A sample output is listed in Figure 2.

## 3 RESULTS

When the NEW protein sequence database of NBRF was searched as described above, 191 proteins were identified to possess at least one Ig domain. A scan of the 4718 proteins comprising the NEW database required an average of 20 hours of CPU time on a VAX 11/780. This is comparable to other computationally intensive programs (e.g., Profile Analysis). When run on a SUN 4 computer, similar searches required 1.3 hours of CPU time. This is sufficiently fast to allow the user to alter the cutoff threshold repeatedly when searching for proteins with low homology.

**Table 1:** Output from a search of the NEW protein sequence database. Domains are sorted according to overall score.

3.0087 Class II histocompatib. antigen, HLA-DR beta-I chain precursor (REM) - Human
3.0148 Nonspecific cross-reacting antigen precursor - Human
3.0161 Platelet-derived growth factor receptor precursor - Mouse
3.0164 Tla class I histocompatib. antigen, T13-c alpha chain - Mouse
3.0164 Tla class I histocompatib. antigen, T3-b alpha chain - Mouse
3.0223 Vitronectin receptor alpha chain precursor - Human
3.0226 T-cell surface glycoprotein Ly-3 precursor - Mouse
3.0244 Kinase-related transforming protein (src) (EC 2.7.1.-) - Avian sarcoma virus
3.0350 Ig alpha-I chain C region - Human
3.0350 Ig alpha-I chain C region - Human
3.0350 Ig alpha-2 chain C region, A2m(I) allotype - Human
3.0409 Granulocyte-macrophage colony-stimulating factor I precursor - Mouse
3.0481 HLA class I histocompatib. antigen, alpha chain precursor - Human
3.0492 NADH-ubiquinone oxidoreductase (EC 1.6.5.3), chain 5 - Fruit fly (Drosophila)
3.0508 NADH-ubiquinone oxidoreductase (EC 1.6.5.3), chain 1 - Fruit fly (Drosophila)
3.0518 HLA class II histocompatib. antigen, DP beta chain precursor - clone
3.0518 HLA class II histocompatib. antigen, DP4 beta chain precursor - Human
3.0518 HLA class II histocompatib. antigen, DPw4 beta-I chain precursor - Human
3.0520 Class II histocompatib. antigen, HLA-DQ beta chain precursor (REM) - Human
3.0561 Protein-tyrosine kinase (EC 2.7.1.112), lymphocyte - Mouse
3.0669 H-2 class II histocompatib. antigen, A-beta-2 chain precursor - Mouse
3.0723 T-cell receptor gamma chain precursor V region (MNG8) - Mouse
3.0723 T-cell receptor gamma chain precursor V region (RACII) - Mouse
3.0723 T-cell receptor gamma chain precursor V region (RAC4) - Mouse
3.0723 T-cell receptor gamma chain precursor V region (RAC42) - Mouse
3.0723 T-cell receptor gamma chain precursor V region (RAC50) - Mouse
3.0750 T-cell receptor beta chain V region (CF6) - Mouse
3.0760 Ig heavy chain V region - Mouse 251.3
3.0781 T-cell receptor beta chain V region (SUP-T1) - Human
3.0787 H-2 class I histocompatib. antigen, Q7 alpha chain precursor - Mouse
3.0787 H-2 class I histocompatib. antigen, Q8 alpha chain precursor - Mouse
3.0982 Myelin-associated glycoprotein 1B236 long form precursor - Rat
3.0982 Myelin-associated glycoprotein 1B236 short form precursor - Rat
3.0982 Myelin-associated glycoprotein precursor, brain - Rat
3.0982 Myelin-associated large glycoprotein precursor - Rat
3.0998 Class I histocompatib. antigen, BoLA alpha chain precursor (8L3-6) - Bovine
3.0998 Class I histocompatib. antigen, BoLA alpha chain precursor (8L3-7) - Bovine
3.1048 H-2 class I histocompatib. antigen, K-k alpha chain precursor - Mouse
3.1086 Ig heavy chain precursor V region - Mouse VGAM3 2
3.1128 T-cell receptor alpha chain precursor V region (MD13) - Mouse
3.1129 T-cell receptor delta chain V region (DN-4) - Mouse
3.1192 T-cell receptor beta chain precursor V region (VAK) - Mouse
3.1265 T-cell receptor gamma chain precursor V region (K20) - Human
3.1347 T-cell receptor alpha chain precursor V region (HAP05) - Human
3.1623 T-cell surface glycoprotein CD8 precursor - Human
3.1623 T-cell surface glycoprotein CD8 protein precursor - Human
3.1776 Ig gamma-3 chain C region, G3m(b) allotype - Human
3.1931 Hypothetical protein HQLF2 - Cytomegalovirus (strain AD169)
3.2041 Sodium channel protein II - Rat
3.2044 Ig heavy chain V region - African clawed frog
3.2147 SURF-1 protein - Mouse
3.2207 T-cell receptor alpha chain precursor v region (HAP10) - Human
3.2300 Beta-2-microglobulin precursor - Human
3.2300 Beta-2-microglobulin, modified - Human
3.2306 Pregnancy-specific beta I glycoprotein E precursor - Human
3.2344 IgE Fc receptor alpha chain precursor - Human
3.2420 T-cell surface glycoprotein CD2 precursor - Pat
3.2422 H-2 class II histocompatib. antigen, I-A (NOD) beta chain precursor - Mouse
3.2552 HLA class II histocompatib. antigen, DPw4 alpha I chain precursor - Human
3.2552 HLA class II histocompatib. antigen, S8 alpha chain precursor - Human
3.2654 T-cell surface glycoprotein CD8, 37K chain precursor - Rat
3.2726 Myelin P0 protein - Bovine
3.2814 Ig alpha-I chain C region - Human
3.2814 Ig alpha-I chain C region - Human
3.2820 Thy-I membrane glycoprotein precursor - Mouse
3.2840 Smh class II histocompatib. antigen - Ehrenbergs mole-rat
3.3039 X-linked chronic granulomatous disease protein - Human
3.3083 Pregnancy-specific beta I glycoprotein C precursor - Human
3.3083 Pregnancy-specific beta-I glycoprotein D precursor - Human
3.3084 T-cell receptor beta chain precursor V region (16) - Human
3.3251 Ig gamma-I/Ig gamma-2b Fc receptor precursor - Mouse
3.3414 Hypothetical hybrid Ig/T-cell receptor precursor V region (SUP-T1) - Human
3.3414 Ig heavy chain precursor V II region - Human 71 2
3.3414 Ig heavy chain precursor V II region - Human 71 4
3.3417 Neural cell adhesion protein precursor - Mouse
3.3511 Ig epsilon chain C region - Human
3.3511 Ig epsilon chain C region - Human
3.3522 T-cell receptor alpha chain V region (8DFL alpha I) - Mouse
3.3605 Biliary glycoprotein I - Human
3.3838 T-cell receptor gamma-I chain C region (MNG1 and MNG7) - Mouse
3.3838 T-cell receptor gamma I chain C region - Mouse
3.3861 T-cell gamma chain precursor V region (V3) - Mouse
3.4024 Ig epsilon chain C region - Human
3.4024 Ig epsilon chain C region - Human
3.4110 Ig heavy chain V region - Mouse H36-2
3.4133 Ig heavy chain V region - Mouse H37-60
3.4152 Ig heavy chain V region - Mouse H18-S415
3.4155 Ig kappa chain V region - Mouse HP9
3.4178 Ig heavy chain V region - Mouse IF6
3.4198 Ig kappa chain V region - Mouse HICS 4D1
3.4199 Ig kappa chain V region - Mouse 3D10
3.4199 Ig heavy chain V region - Mouse II CR id 11
3.4211 Ig heavy chain V region - Mouse HP22 and HP27
3.4213 Pregnancy-specific beta-I glycoprotein C precursor - Human
3.4213 Pregnancy-specific beta-I glycoprotein D precursor - Human
3.4218 T-cell receptor beta chain precursor v region (4 C3) - Mouse
3.4218 T-cell receptor beta chain precursor v region (810) - Mouse
3.4282 Sodium channel protein II - Rat
3.4295 Ig kappa chain V region (H28-A2)   Mouse H28-A2
3.4295 Ig kappa chain V region - Mouse H1S8 89H4
3.4295 Ig kappa chain V region   Mouse H37 311
3.4295 Ig kappa chain V region   Mouse H37 40
3.4295 Ig kappa chain V region   Mouse H37 43
3.4295 Ig kappa chain V region   Mouse H37 45

3.4295 Ig kappa chain V region - Mouse H37-80
3.4295 Ig kappa chain V region - Mouse H37-84
3.4295 Ig kappa chain V regions - Mouse H3S-C6 and H220-25
3.4338 T-cell receptor alpha chain precursor V region (P71) - Mouse
3.4572 T-cell surface glycoprotein CD3 epsilon chain - Human
3.4594 T-cell surface glycoprotein CD8 precursor - Mouse
3.4594 T-cell surface glycoprotein Lyt-2 precursor - Mouse
3.4595 T-cell receptor alpha chain precursor V region (HAP08) - Human
3.4606 T-cell receptor gamma-2 chain C region (MNG8 and MNG9) - Mouse
3.4614 T-cell receptor gamma chain C region (PEER) - Human
3.4614 T-cell receptor gamma-I chain C region - Human
3.4614 T-cell receptor gamma-2 chain C region - Human
3.4620 Ig heavy chain V region - Mouse H146-24B3
3.4620 Ig heavy chain V region - Mouse H1S8-89H4
3.4620 Ig heavy chain V region - Mouse H3S-C6
3.4620 Ig heavy chain precursor V region - Mouse MAK33
3.4690 T-cell receptor beta-I chain C region - Human
3.4690 T-cell receptor beta-I chain C region - Mouse
3.4690 T-cell receptor beta-2 chain C region - Human
3.4690 T-cell receptor beta-2 chain C region - Human
3.4769 Ig gamma-3 chain C region, G3m(b) allotype - Human
3.4798 Ig kappa chain V region - Mouse H146-2483
3.4798 Ig kappa chain V region - Mouse H36-2
3.4798 Ig kappa chain V region - Mouse H37-62
3.4798 Ig kappa chain V region - Mouse H37-82
3.4810 Ig kappa chain V-I region - Human Wil(1)
3.4840 Peroxidase (EC 1.11.1.7) precursor - Human
3.4888 Platelet-derived growth factor receptor precursor - Mouse
3.4965 Notch protein - Fruit fly
3.4965 Notch protein - Fruit fly
3.4983 T-cell receptor beta chain precursor V region (MT I-I) - Human
3.4983 T-cell receptor beta-2 chain precursor V region MOLT-4 - Human
3.4998 Ig kappa chain precursor V region - Mouse Ser-a
3.5035 Alkaline phosphatase (EC 3.1.3.1) precursor - Human
3.5061 Ig heavy chain V region - Mouse H37-82
3.5082 Class II histocompatib. antigen, HLA-DR beta-2 chain precursor (REM) - Human
3.5082 H-2 class II histocompatib. antigen, E-a/k beta-2 chain precursor - Mouse
3.5082 H-2 class II histocompatib. antigen, E-d beta-2 chain precursor - Mouse
3.5082 HLA class II histocompatib. antigen, DR I beta chain (clone 69) - Human
3.5082 HLA class II histocompatib. antigen, DR beta chain precursor
3.5082 HLA class II histocompatib. antigen, DR beta chain precursor AS) - Human
3.5082 HLA class II histocompatib. antigen, DR-I beta chain precursor - Human
3.5082 HLA class II histocompatib. antigen, DR-4 beta chain - Human
3.5082 HLA class II histocompatib. antigen, DR-S beta chain precursor - Human
3.5094 Ig lambda-S chain C region - Mouse
3.5144 Ig alpha-2 chain C region, A2m(1) allotype - Human
3.5150 Ig heavy chain V region - Mouse H28-A2
3.5180 Biliary glycoprotein I - Human
3.5193 Ig heavy chain V region - Mouse H37-45
3.5193 Ig heavy chain V regions - Mouse H37-80 and H37-43
3.5211 Ig lambda chain precursor V region - Rat
3.5264 Ig heavy chain V region - Mouse H37-62
3.5316 Ig heavy chain V region - Mouse H37-311
3.5334 Ig heavy chain V region - Mouse H37-40
3.5372 T-cell receptor beta chain precursor V region (ATL12-2) - Human
3.5435 Ig heavy chain V region - Mouse HICS-40I
3.5579 Ig heavy chain V region - Mouse H37-84
3.5603 Ig lambda-2 chain C region - Rat
3.5666 Ig heavy chain V region - Mouse 81-8 (tentative sequence)
3.5709 Biliary glycoprotein I - Human
3.5748 Nonspecific cross-reacting antigen precursor - Human
3.5815 Ig epsilon chain C region - Human
3.5815 Ig epsilon chain C region - Human
3.5894 Neural cell adhesion protein precursor - Mouse
3.5912 Ig kappa chain V region - Mouse H37-60
3.5971 Ig kappa chain precursor V region - Rat IR2
3.6020 Ig kappa chain V region - Mouse IF6
3.6020 Ig kappa chain V region - Mouse 3D10
3.6027 T-cell receptor beta chain V region (KO-ATL) - Human
3.6071 Ig heavy chain V region - Mouse HP20
3.6071 Ig heavy chain V region - Mouse HP25
3.6120 T-cell receptor alpha chain V region (SCC7) - Mouse
3.6120 T-cell receptor alpha chain V region (CF6) - Mouse
3.6120 T-cell receptor alpha chain precursor V region (2B4) - Mouse
3.6120 T-cell receptor alpha chain precursor V region (4C3) - Mouse
3.6120 T-cell receptor alpha chain precursor V region (810) - Mouse
3.6302 HLA class II histocompatib. antigen DX alpha chain precursor - Human
3.6302 HLA class II histocompatib. antigen, DQ alpha chain precursor - Human
3.6461 T-cell receptor alpha chain precursor V region (HAPS8) - Human
3.6465 Ig kappa chain precursor V chain - Mouse Ser-b
3.6539 Neural cell adhesion protein precursor - Mouse
3.6636 Ig heavy chain V region - Mouse 81-&V1/V2 (tentative sequence)
3.6778 Ig kappa chain precursor V-III region - Human SU-DHL-6
3.6798 Ig kappa chain V region - Mouse H18-S415
3.6803 Myelin-associated glycoprotein 1B236 long form precursor - Rat
3.6803 Myelin-associated glycoprotein 1B236 short form precursor - Rat
3.6803 Myelin-associated glycoprotein precursor, brain - Rat
3.6803 Myelin-associated large glycoprotein precursor - Rat
3.7102 Ig kappa chain V-III region - Human Ger
3.7170 Ig kappa chain V-I region - Human Wil(2)
3.7341 Ig lambda chain C region - Chicken
3.7505 Ig kappa chain precursor V-I region - Human Nalm-6
3.7535 Ig heavy chain precursor V region - Mouse I29
3.7600 Ig lambda-S chain C region - Mouse
3.7779 Ig heavy chain V region - Mouse HP12
3.7907 Ig kappa chain V region 305 precursor - Human
3.7907 Ig kappa chain precursor V-III - Human Nalm-6
3.7909 Ig heavy chain V region - Mouse HP21
3.8087 Neural cell adhesion protein precursor - Mouse
3.8180 Ig mu chain C region, b allele - Mouse
3.8247 Ig epsilon chain C region - Human
3.8247 Ig epsilon chain C region - Human
3.8440 Ig kappa chain precursor V region - Mouse MAK33
3.8678 Ig kappa chain precursor V region - Rat IR162

**Table 2:** Efficiency of detection for some Ig superfamily proteins present in NEW. Mean scores of recognized Ig domains for each protein type are listed. Recognition efficiency is calculated by dividing the number of proteins correctly identified (i.e., bearing at least one Ig domain) by the total number of proteins identified by their file description as containing an Ig domain, multiplied by 100. Numbers in parentheses indicate the number of complete protein sequences of each type for each species. All complete sequences for light and heavy immunoglobulin chains of human and mouse origin were scanned. The threshold was set at 3.0. ND: not done.

| Protein | Mean score of detected domains (max 4.00) | Recognition efficiency for Ig-bearing proteins (see legend) |
|---|---|---|
| Immunoglobulins, mouse, all forms | 3.50 | 98.2 % (55) |
| Immunoglobulins, human, all forms | 3.48 | 93.8 % (16) |
| H-2 class II, all forms | 3.33 | ND |
| HLA class II, all forms | 3.36 | ND |
| T-cell receptor chains, mouse, all forms | 3.32 | ND |
| T-cell receptor chains, human, all forms | 3.41 | ND |

The vast majority of proteins which scored above 3.0 were of human, mouse, rat or rabbit origin. A few viral and insect proteins also scored above the threshold. All proteins in the training set and present in either the NEW or PROTEIN databases were detected. Proteins detected in the NEW database are listed in Table I and sorted according to score. Even though only human MHC class I and II were included in the training set, both mouse H-2 class I and II were detected. Bovine and rat transplantation antigens were also detected. These proteins are homologs of human MHC's. For proteins which include more than one Ig domain contiguously arranged (e.g., carcinoembryonic antigen), all domains were detected if they were sufficiently well conserved. However, domains lacking a feature or possessing a degenerate feature scored much lower (usually below 3.0) such that they are not recognized when using a threshold value of 3. Recognition of human and mouse immunoglobulin sequences was used to measure recognition efficiency. The rate of false negatives for immunoglobulins was very low for both species (Table II). Table III lists the 13 proteins categorized as false positives detected when searching with a threshold of 3.0. Relative to the total number of domains detected, this corresponds to a false positive rate of 6.8%. In the strict sense some of these proteins are not false positives because they do exhibit the expected features of the Ig domain in the correct order. However, inter-feature

distances for these pseudo-domains are very different from those observed in bona fide Ig domains. Proteins which are rich in $\beta$-sheets, such as rat sodium channel II and fruit-fly NADH-ubiquinone oxidoreductase chain 1 are also abundant among the set of false positives. This is not surprising since the Ig domain is composed of $\beta$-strands. One solution to this problem lies in the use of a larger training set as well as the addition of a more intelligent second stage designed to evaluate inter-feature distances so as to increase the specificity of detection.

**Table 3:** False positives obtained when searching NEW with a threshold of 3.0. Proteins categorized as false positives are listed. See text for details.

| |
|---|
| 3.0244 Kinase-related transforming protein (src) (EC 2.7.1.-) |
| 3.0409 Granulocyte-macrophage colony-stimulating |
| 3.0492 NADH-ubiquinone oxidoreductase (EC 1.6.5.3), chain 5 |
| 3.0508 NADH-ubiquinone oxidoreductase (EC 1.6.5.3), chain 1 |
| 3.0561 Protein-tyrosine kinase (EC 2.7.1.112), lymphocyte - Mouse |
| 3.1931 Hypothetical protein HQLF2 - Cytomegalovirus (strain AD169) |
| 3.2041 Sodium channel protein II - Rat |
| 3.2147 SURF-1 protein - Mouse |
| 3.3039 X-linked chronic granulomatous disease protein - Human |
| 3.4840 Peroxidase (EC 1.11.1.7) precursor - Human |
| 3.4965 Notch protein - Fruit fly |
| 3.4965 Notch protein - Fruit fly |
| 3.5035 Alkaline phosphatase (EC 3.1.3.1) precursor - Human |

# 5  DISCUSSION

The detection of specific protein domains is becoming increasingly important since many proteins are constituted of a succession of domains. Unfortunately, domains (Ig or otherwise) are often only weakly homologous with each other. We have designed a neural network to detect proteins which comprise Ig domains to evaluate this approach in helping to solve this problem. Alternatives to neural network-based search programs exist. Search programs can be designed to recognize the flanking Cys-termini regions to the exclusion of other domain features since these flanks are the best conserved features of Ig domains (*cf.* Wang *et al.*, 1989). However, even Cys-termini can exhibit poor overall homology and therefore generate statistically insignificant homology scores when analyzed with the ALIGN program (NBRF) (cf. Williams and Barclay, 1987). Other search programs (such as Profile Analysis) cannot efficiently handle the large variations in domain size exhibited by the Ig domain (mostly comprised between 45 and 70 residues). Search results become corrupted by high rates of false positives and negatives. Since the size of the NBRF protein databases increases considerably each year, the problem of false positives promises to become crippling if these rates are not substantially decreased. In view of these problems we have found the application of a neural network to the detection of Ig domains to be an advantageous solution. As the state of biological knowledge advances, new Ig domains can be added to the training set and training resumed. They can learn the statistical features

of the conserved subregions that permit detection of an Ig domain and generalize to new examples of this domain that have a similar distribution. Previously unrecognized and possibly degenerate homologous sequences are therefore likely to be detected.

## Acknowledgments

This research was supported by a grant from the Canadian Natural Sciences and Engineering Research Council to Y.B. We thank CISTI for graciously allowing us access to their experimental BIOMOLE system.

## References

Bengio Y., Cardin R., De Mori R., Merlo E. (1989) Programmable execution of multi-layered networks for automatic speech recognition, *Communications of the Association for Computing Machinery*, 32 (2).

Bengio Y., Cardin R., De Mori R., (1990), Speaker independent speech recognition with neural networks and speech knowledge, in D.S. Touretzky (ed.), *Advances in Neural Networks Information Processing Systems 2*

Blaschuk O.W., Pouliot Y., Holland P.C., (1990). Identification of a conserved region common to cadherins and influenza strain A hemagglutinins. *J. Molec. Biology*, 1990, in press.

Devereux, J., Haeberli, P. and Smithies, O. (1984) A comprehensive set of sequence analysis programs for the VAX. *Nucl. Acids Res.* 12, 387-395.

Gribskov, M., McLachlan, M., and Eisenber, D. (1987) Profile analysis: Detection of distantly related proteins. *Proc. Natl. Acad. Sci.* USA, 84:4355-4358.

Needleman, S. B. and Wunsch, C. D. (1970) A general method applicable to the search for similarities in the amino acid sequence of two proteins. *J. Mol. Biol.* 48, 443-453.

Qian, N. and Sejnowski, T. J. (1988) Predicting the secondary structure of globular proteins using neural network models. *J. Mol. Biol.* 202, 865-884.

Rumelhart D.E., Hinton G.E. & Williams R.J. (1986) Learning internal representation by error propagation. *Parallel Distributed Processing* , Vol. 1, MIT Press, Cambridge, pp. 318-362.

Smith, T. F. and Waterman, W. S. (1981). Identification of common molecular subsequences. *J. Mol. Biol. 147* , 195-197.

Stormo, G. D., Schneider, T. D., Gold, L. and Ehrenfeucht, A. Use of the "perceptron" algorithm to distinguish translational initiation sites in E. coli. *Nucl. Acids Res. 10* , 2997-3010.

Wang, H., Wu, J. and Tang, P. (1989) Superfamily expands. *Nature*, 337, 514.

Wilbur, W. J. and Lipman, D. J. (1983). Rapid similarity searches of nucleic acids and protein data banks. *Proc. Natl. Acad. Sci.* USA 80, 726-730.

Williams, A. F. and Barclay, N. A. (1988) The immunoglobulin superfamily - domains for cell surface recognition. *Ann. Rev. Immunol.*, 6, 381-405.
